# A four neuron circuit accounts for change sensitive inhibition in salamander retina

Jeffrey L. Teeters
Lawrence Livermore Lab
PO Box 808, L-426
Livermore CA 94550

Frank H. Eeckman
Lawrence Livermore Lab
PO Box 808, L-270
Livermore CA 94550

Frank S. Werblin
UC-Berkeley
Room 145, LSA
Berkeley CA 94720

## Abstract

In salamander retina, the response of On-Off ganglion cells to a central flash is reduced by movement in the receptive field surround. Through computer simulation of a 2-D model which takes into account their anatomical and physiological properties, we show that interactions between four neuron types (two bipolar and two amacrine) may be responsible for the generation and lateral conductance of this change sensitive inhibition. The model shows that the four neuron circuit can account for previously observed movement sensitive reductions in ganglion cell sensitivity and allows visualization and prediction of the spatio-temporal pattern of activity in change sensitive retinal cells.

## 1 INTRODUCTION

In the salamander retina, the response of transient (On-Off) ganglion cells to a central flash is reduced by movement in the receptive field surround (Werblin, 1972; Werblin & Copenhagen, 1974) as illustrated in Fig 1. This phenomenon requires the detection of change in the surround and the lateral transmission of this change sensitive inhibition to the ganglion cell dendrites. Wunk & Werblin (1979) showed that all ganglion cells receive change-sensitive inhibition, and Barnes & Werblin (1987) implicated a change-sensitive amacrine cell with widely distributed processes. The change-sensitivity of these amacrine cells has been traced in part to a truncation of synaptic release from the bipolar terminals that presumably drive them (Maguire et al., 1989). The transient response of these amacrine cells, mediated by voltage gated currents (Barnes & Werblin, 1986; Eliasof et al., 1987) also contributes to this change sensitivity.

These and other experiments suggest that interactions between four neuron types underlie both the change detection and the lateral transmission of inhibition (Werblin et al., 1988; Maguire et al., 1989). To test this hypothesis and make predictions that could be compared with later experiments we have constructed a computational model of the four neuron circuit and incorporated it into an overall model of the retina. This model allows us to simulate the effect of inhibition generated by the four neuron circuit on ganglion cells.

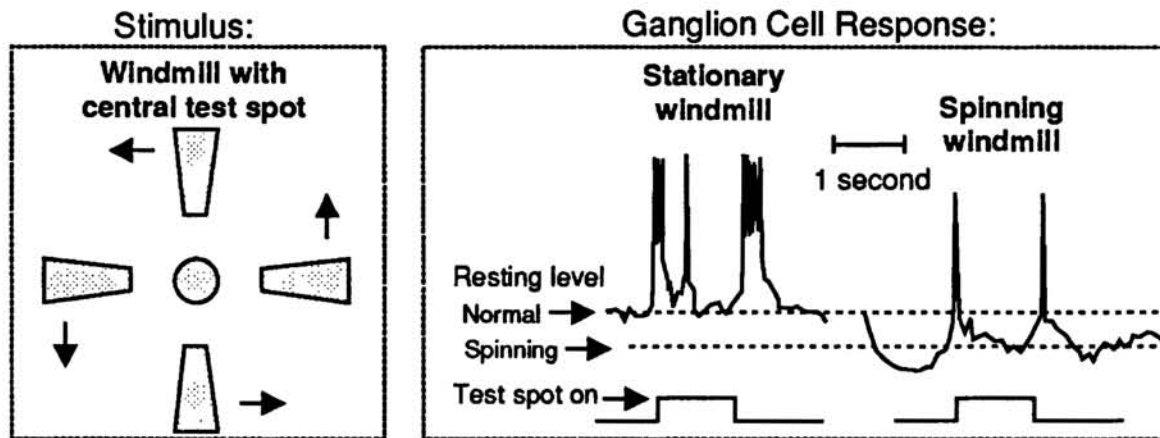

Figure 1: Change-Sensitive Inhibition. Data is from Werblin (1972).

## 2 IMPLEMENTING THE HYPOTHETICAL CIRCUIT

The proposed change-sensitive circuit (Werblin *et al.*, 1988; Maguire *et al.*, 1989) is reproduced in Figure 2. This is meant to describe a very local region of the retina where the receptive fields of the two bipolar cells are spatially overlapping. When a visual target enters this receptive field, the bipolar cells are both depolarized. The sustained bipolar cell activates the narrow field amacrine cell that, in turn feeds back to the synaptic terminal of the transient bipolar cell to truncate transmitter release after a brief (ca. 100 msec) delay. Because the signal reaching the wide field amacrine cell is truncated after about 100 msec, the wide field amacrine cell will receive excitation when the target enters the receptive field, but will not continue to respond in the presence of the target.

The spatial profiles of synaptic input and output for the cell types involved in the model are summarized in Figure 3. The bipolar and narrow field amacrine cell sensitivities extend over a region corresponding roughly to their dendritic spread. The wide field amacrine cell appears to receive input over a local region near the cell body, but delivers its inhibitory output over a much wider region corresponding the the full extent (ca. 500 mm) of its processes.

Figure 4 shows the electrical circuit model for each cell type, and illustrates the interactions between cells that are implemented in the model. In Figure 4, boxes contain the circuit for each cell and arrows between them represent synaptic interactions thought

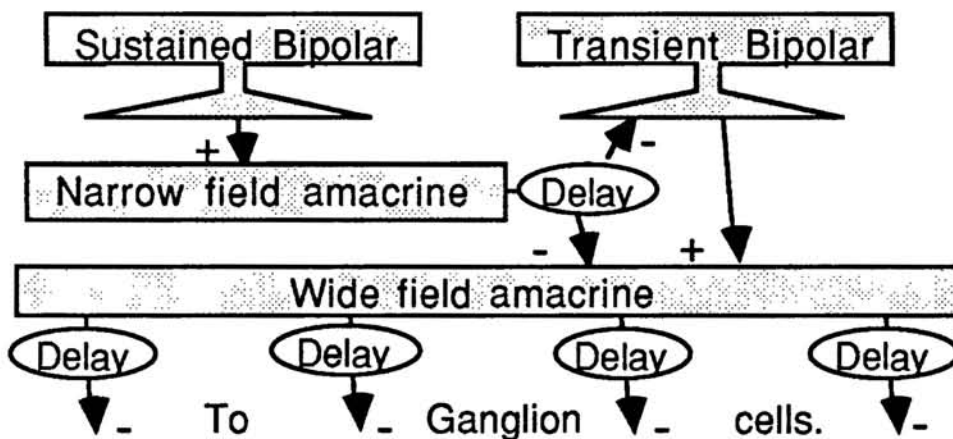

Figure 2: Circuitry to be Analyzed

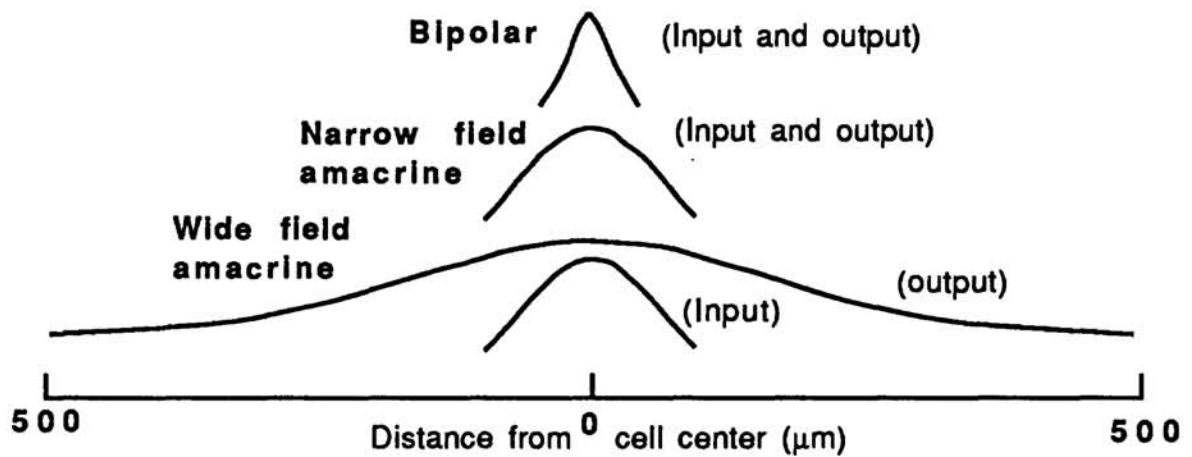

**Bipolar** (Input and output)

**Narrow field amacrine** (Input and output)

**Wide field amacrine** (output)

(Input)

500 ⊢ Distance from 0 cell center (μm) ⊣ 500

Figure 3: Spatial Profiles of Input Sensitivity and Output Transmission

to occur as determined through experiments in which a neurotransmitter is puffed onto bipolar dendrites. Bipolar cells are modeled using two compartments, corresponding to the cell body and axon terminal as suggested in Maguire *et al.* (1989). Amacrine cells are modeled using only one compartment as in Eliasof *et al.* (1987).

Each compartment has a voltage (Vbs, Vbst, Vbtt, Van, Vaw). The cell body for the sustained and transient bipolar are assumed to be the same. Batteries in the figure correspond to excitatory (E+, Ena) or inhibitory reversal potentials (E-, Ek, Ecl). Resistors represent ionic conductances. Circles and arrows through resisters indicate transmitter dependent conductances which are controlled by the voltage of a presynaptic or same cell. Functions relating voltages to conductances are mostly linear with a threshold. More details are given in Teeters *et al.* (1991).

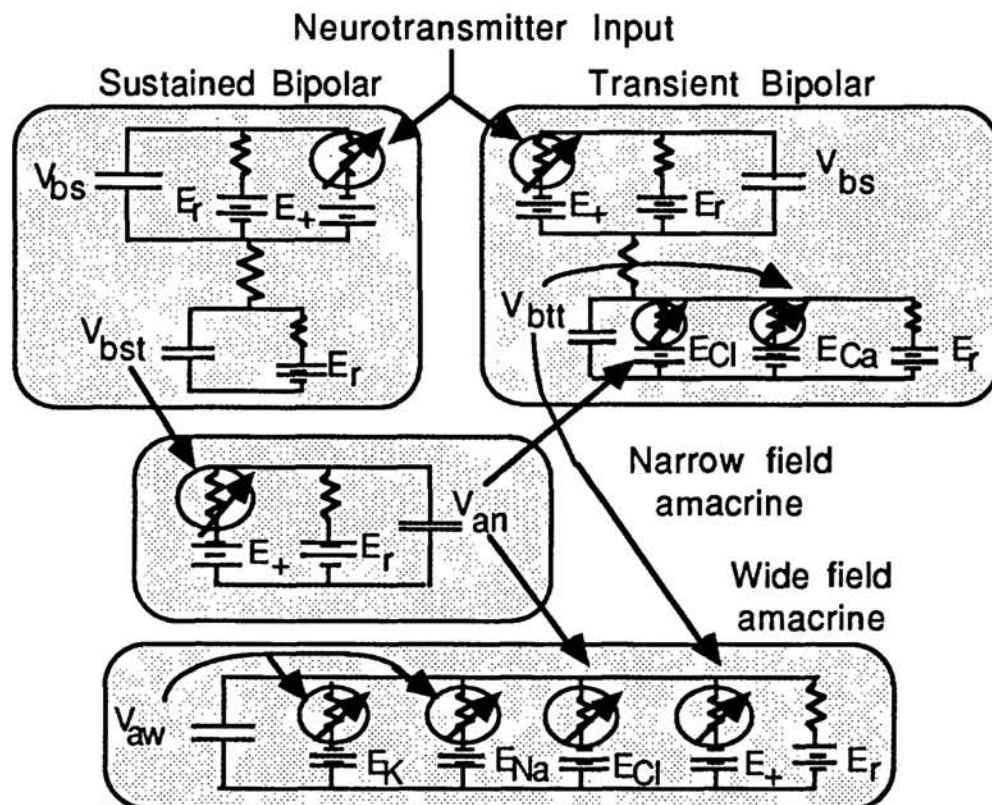

Figure 4: Details of Circuitry

# 3  TESTING THE COMPUTATIONAL MODEL

Computer simulation was used to tune model parameters, and test whether the single cell properties and proposed interactions between cells shown in Figure 4 are consistent with the responses recorded from the neurons during applications of a neurotransmitter puff.

Results are shown in Figure 5. Voltage clamp experiments electrically clamp the cell membrane potential to a constant voltage and determine the current required to maintain the voltage over time. Downward traces indicate that current is flowing into the cell; upward traces indicate outward current. For simplicity, scales are not shown, but in all cases the magnitude of the simulated response is close to that of the observed response.

The simulated and observed responses voltage clamps of the wide field amacrine shown in the fourth row vary because there is a sustained outward current observed experimentally that is not apparent in the simulations. This shows that the model is not perfect and is something that needs further investigation.

This difference between the model and observed response does not prevent the hypothesized function of the circuit from being simulated. This is shown on the bottom row where both the observed and simulated voltage responses from the wide field amacrine are transient.

# 4  SIMULATING INHIBITION TO GANGLION CELLS

Figure 5 illustrates that we have, to a large degree, succeeded in combining the characteristics of single cells into a model which can explain many of the observed properties thought to be due to the interaction between these cells in a local region.

| Experiment | Observed response | Simulated Response |
|---|---|---|
| Neurotransmitter Puff Input | 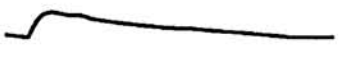 |  |
| Voltage clamp of bipolar cell body |  |  |
| Voltage clamp of narrow field amacrine |  | 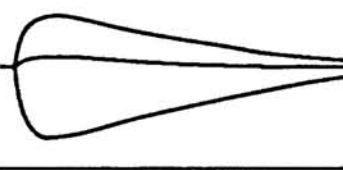 |
| Wide field amacrine *Voltage clamp* | 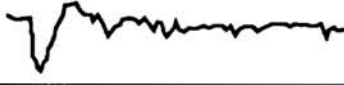 |  |
| *Voltage clamp with picrotoxin block* |  |  |
| *Voltage response* |  | 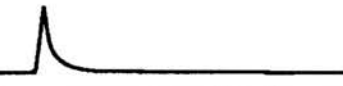 |

Figure 5: Example Puff Simulations

The next step in our analysis is to investigate how this circuit influences the response of ganglion cells. To do this requires simulating the input to the bipolar dendrites and simulating the ganglion cells which receive the transient inhibition generated by the wide field amacrine. This amounts to a integrated model of an entire patch of retina, including receptors, horizontal cells, the four neuron circuit discussed earlier, and ganglion cells. The manner in which we accomplish this is illustrated in Figure 6.

The left side of figure 6 shows the model elements. Receptors and horizontal cells are modeled as low pass filters with different time constants and different spatial inputs. The ganglion cell model receives a transient excitatory input generated phenomenologically by a thresholded high pass filter from the transient bipolar. Inhibitory input to the ganglion cell is implemented as coming from the transient wide field amacrine cells described previously. For simplicity, voltage gated currents and spiking are not implemented in the ganglion cell model, and only the off bipolar pathways are simulated.

The right hand of Figure 6 illustrates how the model is implemented spatially. The circuit for each cell type is duplicated across the retina patch in a matrix format. The known spatial properties of each cell, such as the spatial range of transmitter sensitivity and release are incorporated into the model. Details are given in Teeters *et al.* 1991.

## 5  SIMULATING INHIBITION TO GANGLION CELLS

To test if the model can account for the observed reduction in ganglion cell response during movement in the receptive field surround, we simulated the experiment depicted in Figure 1, mainly the flashing of a central light during the presence of a stationary and spinning windmill. The results are shown in Figure 7.

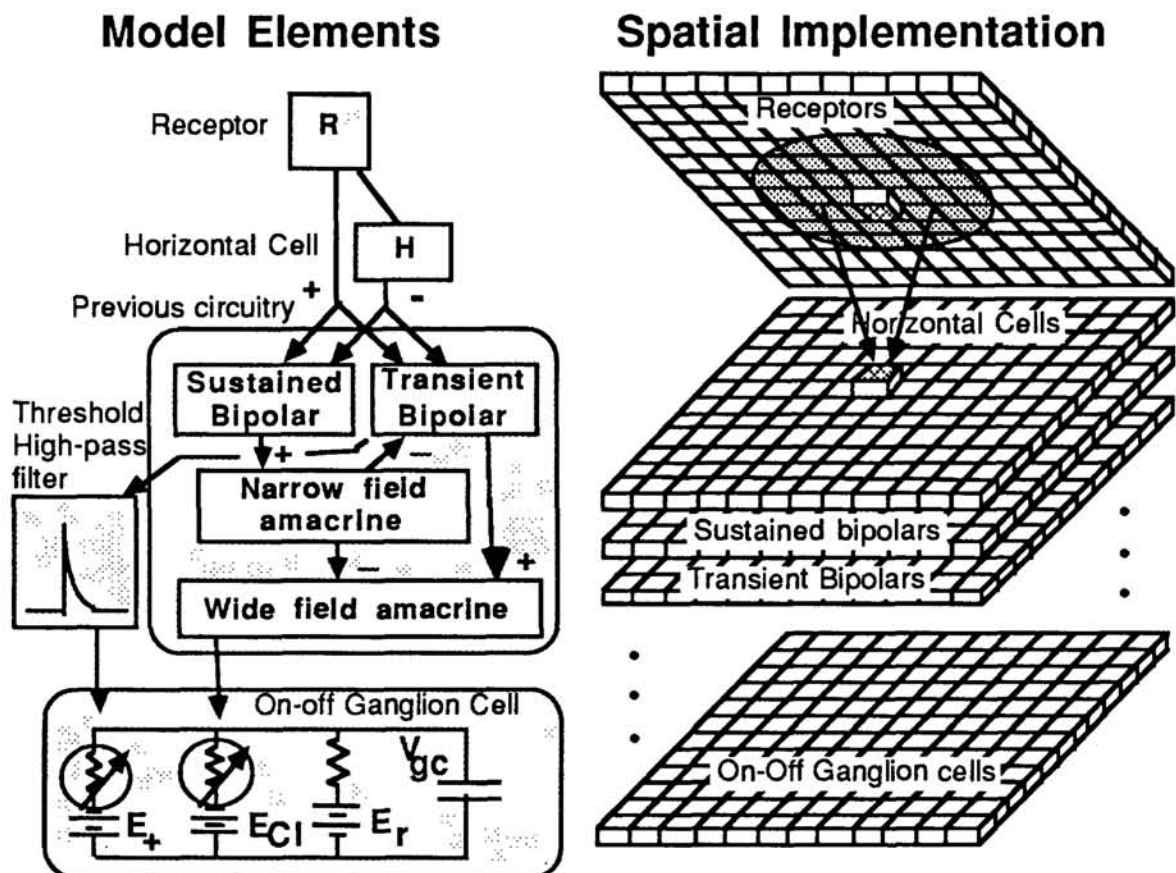

Figure 6: Integrated Retinal Model

Rather than displaying a single curve representing the response of a single unit over time, Figure 7 shows the simultaneous pattern of activity in an array of neurons spatially distributed across the retina patch at an instant in time (just after a central light spot is turned on).  The neuron responses are the transient bipolar terminal, the wide field amacrine neurotransmitter release, and the ganglion cell voltage response.  On the left column is shown the response to a flashing spot when the windmill is stationary.  On the right is shown the response to the same flashing spot but with a spinning windmill.

When the windmill is stationary, the transient bipolar terminal responds only to the center flash.  Responses to the windmill vanes are suppressed by the narrow field amacrine cell causing the appearance of four regions of hyperpolarizing responses around the center.  The wide field amacrine responds to the central test flash and releases transmitter as shown in the second row.  The array of ganglion cells responds to both the excitatory input generated by the spot at the bipolar terminals and the inhibitory input generated by the wide field amacrines.  Because the wide field inhibition has not yet taken effect at this point in time, the ganglion cells respond well to the flashing spot.

When the windmill is spinning, as is shown on the right hand column, the transient bipolar terminals generate a response to the leading edge of the windmill vanes.  The wide field amacrine cells receive excitatory input from the transient bipolar terminal responses to the vane, and consequently release inhibitory neurotransmitter over a wide area as shown in in the right column.  Because inhibition is being continuously generated by the spinning windmill, the response of the ganglion cells across the retinal patch has a large

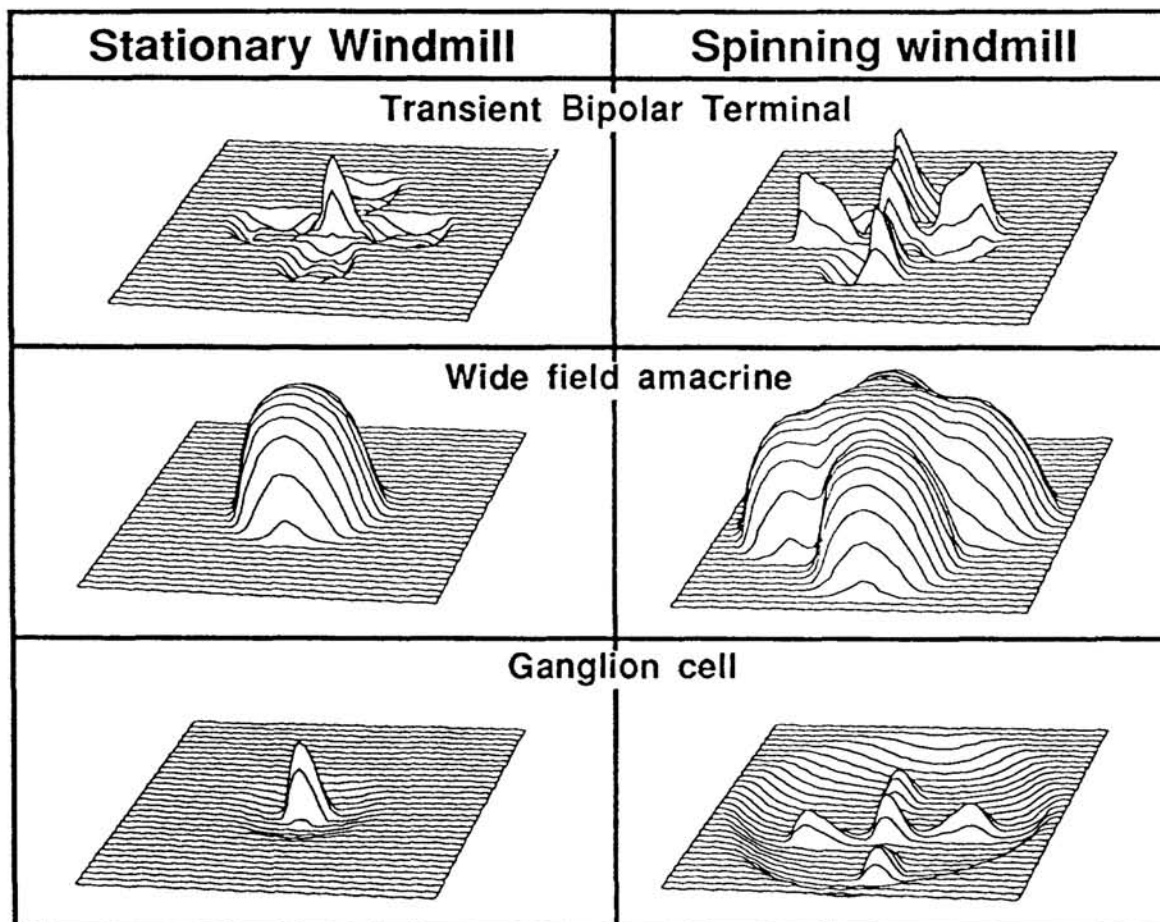

Fig. 7 - Ganglion Cell Inhibition Caused By Spinning Windmill

bowl shaped area of hyperpolarization which reduces the ganglion cell response of the cells to the central test flash. This is seen by the fact that the height of depolarization in the centrally located ganglion cells is much smaller under conditions of a spinning windmill than if the windmill is stationary. This is consistent with the results found experimentally which are illustrated in Figure 1. Experimental data not yet attained, but which are predicted by the model simulations illustrated in Figure 7, are the spatial patterns of activity generated in the bipolar, amacrine, and ganglion cells in response to the different stimuli.

# 6  SUMMARY

Using computer simulation of a neurophysiologically based model, we demonstrate that the experimental data describing properties of four neurons in the inner retina are compatible with the hypothesis that these neurons are involved in the detection of change and the feedforward of change-sensitive inhibition to ganglion cells. First, we build a computational model of the hypothesized four neuron circuit and determine that the proposed interactions between them are sufficient to reproduce many of the observed network properties in response to a puff of neurotransmitter. Next, we integrate this model into a full retina model to simulate their influence on ganglion cell responses.

The model verifies the consistency of presently available data, and allows formation of predictions of neural activity are subject to refutation or verification by new experiments. We are currently recording the spatio-temporal response of ganglion cells to moving stimuli so that direct comparisons to these model predictions can be made.

# References

Barnes, S. and Werblin, F.S. (1986). Gated currents generate single spike activity in amacrine cells of the tiger salamander. *Proc. Natl. Acad. Sci. USA* 83: 1509 - 1512.

Barnes, S. and Werblin, F.S. (1987). Direct excitatory and lateral inhibitory synaptic inputs to amacrine cells in the tiger salamander retina. *Brain Res.* 406: 233 - 237.

Eliasof S., Barnes S. and Werblin, F.S. (1987). The interaction of ionic currents mediating single spike activity in retinal amacrine cells of the tiger salamander. *J. Neurosci.* 7: 3512 - 3524.

Maguire, G., Lukasiewicz, P. and Werblin F.S. (1989). Amacrine cell interactions under-lying the response to change in the tiger salamander retina. *J. Neurosci.* 9: 726 - 735.

Teeters, J.L., Eeckman, F.H., Werblin F.S. (1991). A computer model to visualize change sensitive responses in the salamander retina. In MA. Arbib and J-P. Ewert (eds.) *Visuomotor Coordination: Amphibians, Comparisons, Models and Robots.* Plenum.

Werblin, F.S. (1972). Lateral interactions at inner plexiform layer of a vertebrate retina: antagonistic response to change. *Science.* 175: 1008 - 1010.

Werblin, F.S. and Copenhagen, D.R. (1974). Control of retinal sensitivity. III. Lateral interactions at the inner plexiform layer. *J. Gen. Physiol.* 63: 88 - 110.

Werblin, F.S., Maguire, G., Lukasiewicz, P., Eliasof, S., and Wu, S. (1988). Neural interactions mediating the detection of motion in the retina of the tiger salamander. *Visual Neurosci.* 1: 317 - 329.

Wunk, D.F. and Werblin, F.S. (1979). Synaptic inputs to ganglion cells in the tiger salamander retina. *J. Gen. Physiol.* 73: 265 - 286.
